# Incremental A*

**S. Koenig and M. Likhachev**
Georgia Institute of Technology
College of Computing
Atlanta, GA 30312-0280
{ *skoenig, mlikhach* }*@cc.gatech.edu*

## Abstract

Incremental search techniques find optimal solutions to series of similar search tasks much faster than is possible by solving each search task from scratch. While researchers have developed incremental versions of uninformed search methods, we develop an incremental version of A*. The first search of Lifelong Planning A* is the same as that of A* but all subsequent searches are much faster because it reuses those parts of the previous search tree that are identical to the new search tree. We then present experimental results that demonstrate the advantages of Lifelong Planning A* for simple route planning tasks.

## 1  Overview

Artificial intelligence has investigated knowledge-based search techniques that allow one to solve search tasks in large domains. Most of the research on these methods has studied how to solve one-shot search problems. However, search is often a repetitive process, where one needs to solve a series of similar search tasks, for example, because the actual situation turns out to be slightly different from the one initially assumed or because the situation changes over time. An example for route planning tasks are changing traffic conditions. Thus, one needs to replan for the new situation, for example if one always wants to display the least time-consuming route from the airport to the conference center on a web page. In these situations, most search methods replan from scratch, that is, solve the search problems independently. Incremental search techniques share with case-based planning, plan adaptation, repair-based planning, and learning search-control knowledge the property that they find solutions to series of similar search tasks much faster than is possible by solving each search task from scratch. Incremental search techniques, however, differ from the other techniques in that the quality of their solutions is guaranteed to be as good as the quality of the solutions obtained by replanning from scratch.

Although incremental search methods are not widely known in artificial intelligence and control, different researchers have developed incremental search versions of uninformed search methods in the algorithms literature. An overview can be found in [FMSN00]. We, on the other hand, develop an incremental version of A*, thus combining ideas from the algorithms literature and the artificial intelligence literature. We call the algorithm Lifelong Planning A* (LPA*), in analogy to "lifelong learning" [Thr98], because it reuses

---

*We thank Anthony Stentz for his support. The Intelligent Decision-Making Group is partly supported by NSF awards under contracts IIS-9984827, IIS-0098807, and ITR/AP-0113881. The views and conclusions contained in this document are those of the authors and should not be interpreted as representing the official policies, either expressed or implied, of the sponsoring organizations and agencies or the U.S. government.

information from previous searches. LPA* uses heuristics to focus the search and always finds a shortest path for the current edge costs. The first search of LPA* is the same as that of A* but all subsequent searches are much faster. LPA* produces at least the search tree that A* builds. However, it achieves a substantial speedup over A* because it reuses those parts of the previous search tree that are identical to the new search tree.

## 2   The Route Planning Task

Lifelong Planning A* (LPA*) solves the following search task: It applies to finite graph search problems on known graphs whose edge costs can increase or decrease over time. $S$ denotes the finite set of vertices of the graph. $Succ(s) \subseteq S$ denotes the set of successors of vertex $s \in S$. Similarly, $Pred(s) \subseteq S$ denotes the set of predecessors of vertex $s \in S$. $0 < c(s, s') \leq \infty$ denotes the cost of moving from vertex $s$ to vertex $s' \in Succ(s)$. LPA* always determines a shortest path from a given start vertex $s_{start} \in S$ to a given goal vertex $s_{goal} \in S$, knowing both the topology of the graph and the current edge costs. We use $g^*(s)$ to denote the start distance of vertex $s \in S$, that is, the length of a shortest path from $s_{start}$ to $s$.

To motivate and test LPA*, we use a special case of these search tasks that is easy to visualize. We apply LPA* to navigation problems in known eight-connected gridworlds with cells whose traversability can change over time. They are either traversable (with cost one) or untraversable. LPA* always determines a shortest path between two given cells of the gridworld, knowing both the topology of the gridworld and which cells are currently blocked. This is a special case of the graph search problems on eight-connected grids whose edge costs are either one or infinity. As an approximation of the distance between two cells, we use the maximum of the absolute differences of their x and y coordinates. This results in consistent heuristics that are for eight-connected grids what Manhattan distances are for four-connected grids.

## 3   Reusing Information from Previous Searches

The graph search problems can be solved with traditional graph-search methods, such as breadth-first search, if they update the shortest path every time some edge costs change. They typically do not take advantage of information from previous searches. The following example, however, shows that this can be advantageous.

Consider the gridworlds of size $20 \times 15$ shown in Figure 1. The original gridworld is shown on top and the changed gridworld is shown at the bottom. The traversability of only a few cells has changed. In particular, three blocked cells became traversable (namely, B3, C5, and D2) and three traversable cells became blocked (namely, A1, A4, D3). Thus, two percent of the cells changed their status but the obstacle density remained the same. The figure shows the shortest paths in both cases, breaking ties towards the north. Note that we assume that one can squeeze through diagonal obstacles. (This is just an artifact of how we generated the underlying graphs from the mazes.) The shortest path changed since one cell on the original shortest path became blocked.

Once the start distances of all cells are known, one can easily trace back a shortest path from the start cell to the goal cell by always greedily decreasing the start distance, starting at the goal cell. This is similar to how A* traces the shortest path back from $s_{goal}$ to $s_{start}$ using the search tree it has constructed. Thus, we only need to determine the start distances. The start distances are shown in each traversable cell of the original and changed gridworlds. Those cells whose start distances in the changed gridworld have changed from the corresponding ones in the original gridworld are shaded gray.

There are two different ways of decreasing the search effort for determining the start distances for the changed gridworld. First, some start distances have not changed and thus need not get recomputed. This is what DynamicSWSF-FP [RR96] does. (DynamicSWSF-

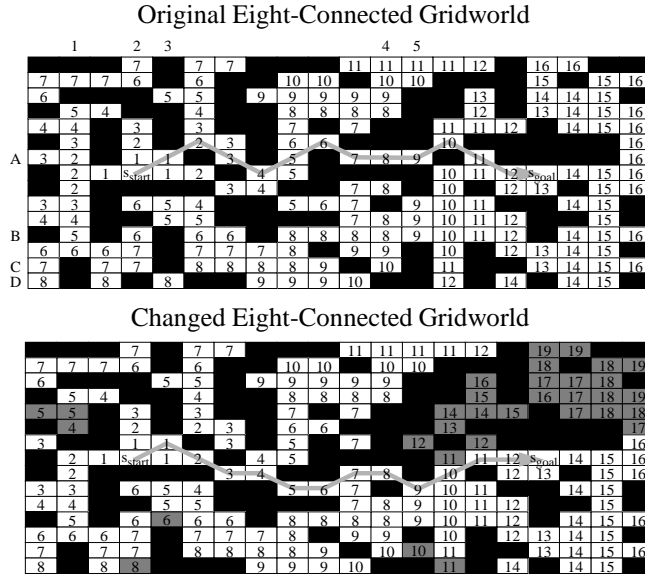

Figure 1: Simple Gridworld

FP, as originally stated, searches from the goal vertex to the start vertex and thus maintains estimates of the goal distances rather than the start distances. It is a simple matter of restating it to search from the start vertex to the goal vertex. Furthermore, DynamicSWSF-FP, as originally stated, recomputes all goal distances that have changed. To avoid biasing our experimental results in favor of LPA*, we changed the termination condition of DynamicSWSF-FP so that it stops immediately after it is sure that it has found a shortest path.) Second, heuristic knowledge, in form of approximations of the goal distances, can be used to focus the search and determine that some start distances need not get computed at all. This is what A* [Pea85] does. We demonstrate that the two ways of decreasing the search effort are orthogonal by developing LPA* that combines both of them and thus is able to replan faster than either DynamicSWSF-FP or A*.

Figure 2 shows in gray those cells whose start distances each of the four algorithms recomputes. (To be precise: it shows in gray the cells that each of the four algorithms expands.) During the search in the original gridworld, DynamicSWSF-FP computes the same start distances as breadth-first search during the first search and LPA* computes the same start distances as A*. During the search in the changed gridworld, however, both incremental search (DynamicSWSF-FP) and heuristic search (A*) individually decrease the number of start distances that need to get recomputed compared to breadth-first search, and together (LPA*) decrease the number even more.

## 4    Lifelong Planning A*

Lifelong Planning A* (LPA*) is an incremental version of A* that uses heuristics $h(s)$ to control its search. As for A*, the heuristics approximate the goal distances of the vertices $s$. They need to be consistent, that is, satisfy $h(s_{goal}) = 0$ and $h(s) \leq c(s, s') + h(s')$ for all vertices $s \in S$ and $s' \in Succ(s)$ with $s \neq s_{goal}$.

LPA* maintains an estimate $g(s)$ of the start distance $g^*(s)$ of each vertex $s$. These values directly correspond to the g-values of an A* search. They are carried forward from search to search. LPA* also maintains a second kind of estimate of the start distances. The rhs-values are one-step lookahead values based on the g-values and thus potentially better informed

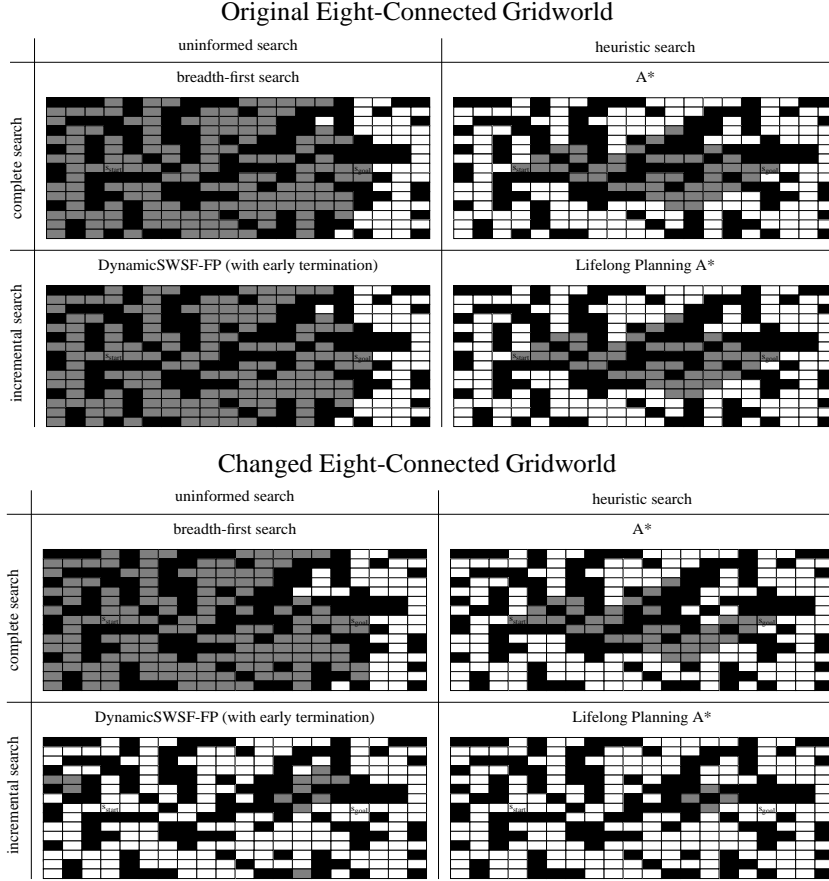

Figure 2: Performance of Search Methods in the Simple Gridworld

than the g-values. They always satisfy the following relationship:

$$rhs(s) = \begin{cases} 0 & \text{if } s = s_{start} \\ \min_{s' \in Pred(s)}(g(s') + c(s', s)) & \text{otherwise.} \end{cases} \quad (1)$$

A vertex is called locally consistent iff its g-value equals its rhs-value. This is similar to satisfying the Bellman equation for undiscounted deterministic sequential decision problems. Thus, this concept is important because the g-values of all vertices equal their start distances iff all vertices are locally consistent. However, LPA* does not make every vertex locally consistent. Instead, it uses the heuristics $h(s)$ to focus the search and update only the g-values that are relevant for computing a shortest path from $s_{start}$ to $s_{goal}$.

LPA* maintains a priority queue $U$ that always contains exactly the locally inconsistent vertices. These are the vertices whose g-values LPA* potentially needs to update to make them locally consistent. The keys of the vertices in the priority queue correspond to the f-values used by A*, and LPA* always expands the vertex in the priority queue with the smallest key, similar to A* that always expands the vertex in the priority queue with the smallest f-value. By expanding a vertex, we mean executing {10-16} (numbers in brackets refer to line numbers in Figure 3). The key $k(s)$ of vertex $s$ is a vector with two components:

The pseudocode uses the following functions to manage the priority queue: U.TopKey() returns the smallest priority of all vertices in priority queue $U$. (If $U$ is empty, then U.TopKey() returns $[\infty;\infty]$.) U.Pop() deletes the vertex with the smallest priority in priority queue $U$ and returns the vertex. U.Insert($s, k$) inserts vertex $s$ into priority queue $U$ with priority $k$. Finally, U.Remove($s$) removes vertex $s$ from priority queue $U$.

**procedure CalculateKey($s$)**
{01} return $[\min(g(s), rhs(s)) + h(s); \min(g(s), rhs(s))]$;

**procedure Initialize()**
{02} $U := \emptyset$;
{03} for all $s \in S$ $rhs(s) = g(s) = \infty$;
{04} $rhs(s_{start}) = 0$;
{05} U.Insert($s_{start}, [h(s_{start}); 0]$);

**procedure UpdateVertex($u$)**
{06} if $(u \neq s_{start})$ $rhs(u) = \min_{s' \in Pred(u)}(g(s') + c(s', u))$;
{07} if $(u \in U)$ U.Remove($u$);
{08} if $(g(u) \neq rhs(u))$ U.Insert($u$, CalculateKey($u$));

**procedure ComputeShortestPath()**
{09} while (U.TopKey() $\dot{<}$ CalculateKey($s_{goal}$) OR $rhs(s_{goal}) \neq g(s_{goal})$)
{10}    $u =$ U.Pop();
{11}    if $(g(u) > rhs(u))$
{12}       $g(u) = rhs(u)$;
{13}       for all $s \in Succ(u)$ UpdateVertex($s$);
{14}    else
{15}       $g(u) = \infty$;
{16}       for all $s \in Succ(u) \cup \{u\}$ UpdateVertex($s$);

**procedure Main()**
{17} Initialize();
{18} forever
{19}    ComputeShortestPath();
{20}    Wait for changes in edge costs;
{21}    for all directed edges $(u, v)$ with changed edge costs
{22}       Update the edge cost $c(u, v)$;
{23}       UpdateVertex($v$);

Figure 3: Lifelong Planning A*.

$$k(s) = [k_1(s); k_2(s)], \tag{2}$$

where $k_1(s) = \min(g(s), rhs(s)) + h(s)$ and $k_2(s) = \min(g(s), rhs(s))$ {1}. Keys are compared according to a lexicographic ordering. For example, a key $k(s)$ is smaller than or equal to a key $k'(s)$, denoted by $k(s) \dot{\leq} k'(s)$, iff either $k_1(s) < k'_1(s)$ or ($k_1(s) = k'_1(s)$ and $k_2(s) \leq k'_2(s)$). $k_1(s)$ corresponds directly to the f-values $f(s) = g^*(s) + h(s)$ used by A* because both the g-values and rhs-values of LPA* correspond to the g-values of A* and the h-values of LPA* correspond to the h-values of A*. $k_2(s)$ corresponds to the g-values of A*. LPA* expands vertices in the order of increasing $k_1$-values and vertices with equal $k_1$-values in order of increasing $k_2$-values. This is similar to A* that expands vertices in the order of increasing f-values (since the heuristics are consistent) and vertices with equal f-values that are on the same branch of the search tree in order of increasing g-values (since it grows the search tree).

A locally inconsistent vertex $s$ is called overconsistent iff $g(s) > rhs(s)$. When LPA* expands a locally overconsistent vertex {12-13}, then $rhs(s) = g^*(s)$ because vertex $s$ has the smallest key among all locally inconsistent vertices. $rhs(s) = g^*(s)$ implies that $k(s) = [f(s); g^*(s)]$ and thus LPA* expands overconsistent vertices in the same order as A*. During the expansion of vertex $s$, LPA* sets the g-value of vertex $s$ to its rhs-value and thus its start distance {12}, which is the desired value and also makes the vertex locally consistent. Its g-value then no longer changes until LPA* terminates. A locally inconsistent vertex $s$ is called underconsistent iff $g(s) < rhs(s)$. When LPA* expands a locally underconsistent vertex {15-16}, then it simply sets the g-value of the vertex to infinity {15}. This makes the vertex either locally consistent or locally overconsistent. If the expanded vertex was locally overconsistent, then the change of its g-value can affect the local consistency of its successors {13}. Similarly, if the expanded vertex was locally underconsistent, then it and its successors can be affected {16}. LPA* therefore updates rhs-values of these vertices, checks their local consistency, and adds them to or removes them from the priority queue accordingly.

LPA* expands vertices until $s_{goal}$ is locally consistent and the key of the vertex to expand next is no smaller than the key of $s_{goal}$. This is similar to A* that expands vertices until it expands $s_{goal}$ at which point in time the g-value of $s_{goal}$ equals its start distance and the f-value of the vertex to expand next is no smaller than the f-value of $s_{goal}$. It turns out that LPA* expands a vertex at most twice, namely at most once when it is underconsistent and at most once when it is overconsistent. Thus, ComputeShortestPath() returns after a number of vertex expansions that is at most twice the number of vertices.

If $g(s_{goal}) = \infty$ after the search, then there is no finite-cost path from $s_{start}$ to $s_{goal}$. Otherwise, one can trace back a shortest path from $s_{start}$ to $s_{goal}$ by always moving from the current vertex $s$, starting at $s_{goal}$, to any predecessor $s'$ that minimizes $g(s') + c(s', s)$ until $s_{start}$ is reached (ties can be broken arbitrarily), similar to what A* can do if it does not use backpointers.

The resulting version of LPA* is shown in Figure 3. The main function Main() first calls Initialize() to initialize the search problem {17}. Initialize() sets the initial g-values of all vertices to infinity and sets their rhs-values according to Equation 1 {03-04}. Thus, initially $s_{start}$ is the only locally inconsistent vertex and is inserted into the otherwise empty priority queue with a key calculated according to Equation 2 {05}. This initialization guarantees that the first call to ComputeShortestPath() performs exactly an A* search, that is, expands exactly the same vertices as A* in exactly the same order, provided that A* breaks ties between vertices with the same f-values suitably. Notice that, in an actual implementation, Initialize() only needs to initialize a vertex when it encounters it during the search and thus does not need to initialize all vertices up front. This is important because the number of vertices can be large and only a few of them might be reached during the search. LPA* then waits for changes in edge costs {20}. If some edge costs have changed, it calls UpdateVertex() {23} to update the rhs-values and keys of the vertices potentially affected by the changed edge costs as well as their membership in the priority queue if they become locally consistent or inconsistent, and finally recalculates a shortest path {19}.

## 5   Optimizations of Lifelong Planning A*

There are several simple ways of optimizing LPA* without changing its overall operation. The resulting version of LPA* is shown in Figure 4. First, a vertex sometimes gets removed from the priority queue and then immediately reinserted with a different key. For example, a vertex can get removed on line {07} and then be reentered on line {08}. In this case, it is often more efficient to leave the vertex in the priority queue, update its key, and only change its position in the priority queue. Second, when UpdateVertex() on line {13} computes the rhs-value for a successor of an overconsistent vertex it is unnecessary to take the minimum over all of its respective predecessors. It is sufficient to compute the rhs-value as the minimum of its old rhs-value and the sum of the new g-value of the overconsistent vertex and the cost of moving from the overconsistent vertex to the successor. The reason is that only the g-value of the overconsistent vertex has changed. Since it decreased, it can only decrease the rhs-values of the successor. Third, when UpdateVertex() on line {16} computes the rhs-value for a successor of an underconsistent vertex, the only g-value that has changed is the g-value of the underconsistent vertex. Since it increased, the rhs-value of the successor can only get affected if its old rhs-value was based on the old g-value of the underconsistent vertex. This can be used to decide whether the successor needs to get updated and its rhs-value needs to get recomputed {21'}. Fourth, the second and third optimization concerned the computations of the rhs-values of the successors after the g-value of a vertex has changed. Similar optimizations can be made for the computation of the rhs-value of a vertex after the cost of one of its incoming edges has changed.

## 6   Analytical and Experimental Results

We can prove the correctness of ComputeShortestPath().

The pseudocode uses the following functions to manage the priority queue: U.Top() returns a vertex with the smallest priority of all vertices in priority queue $U$. U.TopKey() returns the smallest priority of all vertices in priority queue $U$. (If $U$ is empty, then U.TopKey() returns $[\infty; \infty]$.) U.Insert($s$, $k$) inserts vertex $s$ into priority queue $U$ with priority $k$. U.Update($s$, $k$) changes the priority of vertex $s$ in priority queue $U$ to $k$. (It does nothing if the current priority of vertex $s$ already equals $k$.) Finally, U.Remove($s$) removes vertex $s$ from priority queue $U$.

**procedure CalculateKey($s$)**
{01'} return $[\min(g(s), rhs(s)) + h(s); \min(g(s), rhs(s))]$;

**procedure Initialize()**
{02'} $U := \emptyset$;
{03'} for all $s \in S$ $rhs(s) = g(s) = \infty$;
{04'} $rhs(s_{start}) = 0$;
{05'} U.Insert($s_{start}$, $[h(s_{start}); 0]$);

**procedure UpdateVertex($u$)**
{06'} if $(g(u) \neq rhs(u)$ AND $u \in U)$ U.Update($u$, CalculateKey($u$));
{07'} else if $(g(u) \neq rhs(u)$ AND $u \notin U)$ U.Insert($u$, CalculateKey($u$));
{08'} else if $(g(u) = rhs(u)$ AND $u \in U)$ U.Remove($u$);

**procedure ComputeShortestPath()**
{09'} while (U.TopKey() $\dot{<}$ CalculateKey($s_{goal}$) OR $rhs(s_{goal}) \neq g(s_{goal})$)
{10'}    $u = $ U.Top();
{11'}    if $(g(u) > rhs(u))$
{12'}       $g(u) = rhs(u)$;
{13'}       U.Remove($u$);
{14'}       for all $s \in Succ(u)$
{15'}          if $(s \neq s_{start})$ $rhs(s) = \min(rhs(s), g(u) + c(u, s))$;
{16'}          UpdateVertex($s$);
{17'}    else
{18'}       $g_{old} = g(u)$;
{19'}       $g(u) = \infty$;
{20'}       for all $s \in Succ(u) \cup \{u\}$
{21'}          if $(rhs(s) = g_{old} + c(u, s)$ OR $s = u)$
{22'}             if $(s \neq s_{start})$ $rhs(s) = \min_{s' \in Pred(s)}(g(s') + c(s', s))$;
{23'}          UpdateVertex($s$);

**procedure Main()**
{24'} Initialize();
{25'} forever
{26'}    ComputeShortestPath();
{27'}    Wait for changes in edge costs;
{28'}    for all directed edges $(u, v)$ with changed edge costs
{29'}       $c_{old} = c(u, v)$;
{30'}       Update the edge cost $c(u, v)$;
{31'}       if $(c_{old} > c(u, v))$
{32'}          if $(v \neq s_{start})$ $rhs(v) = \min(rhs(v), g(u) + c(u, v))$;
{33'}       else if $(rhs(v) = g(u) + c_{old})$
{34'}          if $(v \neq s_{start})$ $rhs(v) = \min_{s' \in Pred(v)}(g(s') + c(s', v))$;
{35'}       UpdateVertex($v$);

Figure 4: Lifelong Planning A* (optimized version)

**Theorem 1** *ComputeShortestPath() terminates and one can then trace back a shortest path from $s_{start}$ to $s_{goal}$ by always moving from the current vertex $s$, starting at $s_{goal}$, to any predecessor $s'$ that minimizes $g(s') + c(s', s)$ until $s_{start}$ is reached (ties can be broken arbitrarily).*

(The proofs can be found in [LK01].) We now compare breadth-first search, A*, DynamicSWSF-FP, and the optimized version of LPA* experimentally. (We use DynamicSWSF-FP with the same optimizations that we developed for LPA*, to avoid biasing our experimental results in favor of LPA*.) The priority queues of all four algorithms were implemented as binary heaps. Since all algorithms determine the same paths (if they break ties suitably), we need to compare their total search time until a shortest path has been found. Since the actual runtimes are implementation-dependent, we instead use three measures that all correspond to common operations performed by the algorithms and thus heavily influence their runtimes: the total number of vertex expansions $ve$ (that is, updates of the g-values, similar to backup operations of dynamic programming for sequential decision problems), the total number of vertex accesses $va$ (for example, to read or change their values), and the total number of heap percolates $hp$ (exchanges of a parent and child in the heap). Note that we count two vertex expansions, not just one vertex expansion, if LPA* expands the same vertex twice, to avoid biasing our experimental results in favor of LPA*.

All of our experiments use fifty eight-connected gridworlds that have size $40 \times 40$ and an

obstacle density of 40 percent. The start cell is at coordinates (34, 20) and the goal cell is at coordinates (5, 20), where the upper leftmost cell is at coordinates (0, 0). For each gridworld, the initial obstacle configuration was generated randomly. Then, it was changed 500 times in a row, each time by making eight randomly chosen blocked cells traversable and eight randomly chosen traversable cells blocked. Thus, each time one percent of the cells changed their status but the obstacle density remained the same. After each of the 500 changes, the algorithms recomputed a shortest path from the start cell to the goal cell. For each of the four algorithms and each of the three performance measures, the following table reports the mean of the performance measure for the 500 changes: both its average over the fifty mazes and its 95-percent confidence interval over the fifty mazes (assuming a normal distribution with unknown variance). The table confirms the observations made in Section 3: LPA* outperforms the other three search methods according to all three performance measures.

| | uninformed search | | | heuristic search | | |
|---|---|---|---|---|---|---|
| complete search | breadth-first search | | | A* | | |
| | ve = 1331.7 | ± | 4.4 | ve = 284.0 | ± | 5.9 |
| | va = 26207.2 | ± | 84.0 | va = 6177.3 | ± | 129.3 |
| | hp = 5985.3 | ± | 19.7 | hp = 1697.3 | ± | 39.9 |
| incremental search | DynamicSWSF-FP | | | Lifelong Planning A* | | |
| | ve = 173.0 | ± | 4.9 | ve = 25.6 | ± | 2.0 |
| | va = 5697.4 | ± | 167.0 | va = 1235.9 | ± | 75.0 |
| | hp = 956.2 | ± | 26.6 | hp = 240.1 | ± | 16.9 |

We have also applied LPA* successfully to more complex planning tasks, including the kind of route planning tasks that Focussed Dynamic A* [Ste95] applies to. The results will be reported separately.

# References

[FMSN00]  D. Frigioni, A. Marchetti-Spaccamela, and U. Nanni. Fully dynamic algorithms for maintaining shortest paths trees. *Journal of Algorithms*, 34(2):251–281, 2000.

[LK01]  M. Likhachev and S. Koenig. Lifelong Planning A* and Dynamic A* Lite: The proofs. Technical report, College of Computing, Georgia Institute of Technology, Atlanta (Georgia), 2001.

[Pea85]  J. Pearl. *Heuristics: Intelligent Search Strategies for Computer Problem Solving*. Addison-Wesley, 1985.

[RR96]  G. Ramalingam and T. Reps. An incremental algorithm for a generalization of the shortest-path problem. *Journal of Algorithms*, 21:267–305, 1996.

[Ste95]  A. Stentz. The focussed D* algorithm for real-time replanning. In *Proceedings of the International Joint Conference on Artificial Intelligence*, pages 1652–1659, 1995.

[Thr98]  Sebastian Thrun. Lifelong learning algorithms. In S. Thrun and L. Pratt, editors, *Learning To Learn*. Kluwer Academic Publishers, 1998.
